# An experimental comparison of recurrent neural networks

**Bill G. Horne and C. Lee Giles***
NEC Research Institute
4 Independence Way
Princeton, NJ 08540
{horne,giles}@research.nj.nec.com

## Abstract

Many different discrete–time recurrent neural network architectures have been proposed. However, there has been virtually no effort to compare these architectures experimentally. In this paper we review and categorize many of these architectures and compare how they perform on various classes of simple problems including grammatical inference and nonlinear system identification.

## 1 Introduction

In the past few years several recurrent neural network architectures have emerged. In this paper we categorize various discrete–time recurrent neural network architectures, and perform a quantitative comparison of these architectures on two problems: grammatical inference and nonlinear system identification.

## 2 RNN Architectures

We broadly divide these networks into two groups depending on whether or not the states of the network are guaranteed to be observable. A network with observable states has the property that the states of the system can always be determined from observations of the input and output alone. The archetypical model in this class

Table 1: Terms that are weighted in various single layer network architectures. $u_i$ represents the $i^{\text{th}}$ input at the current time step, $z_i$ represents the value of the $j^{\text{th}}$ node at the previous time step.

| Architecture | bias | $u_i$ | $z_i$ | $u_i u_j$ | $z_i u_j$ | $z_i z_j$ |
|---|---|---|---|---|---|---|
| First order | x | x | x | | | |
| High order | | | | | x | |
| Bilinear | | x | x | | x | |
| Quadratic | x | x | x | x | x | x |

was proposed by Narendra and Parthasarathy [9]. In their most general model, the output of the network is computed by a multilayer perceptron (MLP) whose inputs are a window of past inputs and outputs, as shown in Figure 1a. A special case of this network is the Time Delay Neural Network (TDNN), which is simply a tapped delay line (TDL) followed by an MLP [7]. This network is not recurrent since there is no feedback; however, the TDL does provide a simple form of dynamics that gives the network the ability model a limited class of nonlinear dynamic systems. A variation on the TDNN, called the Gamma network, has been proposed in which the TDL is replaced by a set of cascaded filters [2]. Specifically, if the output of one of the filters is denoted $x_j(k)$, and the output of filter $i$ connects to the input of filter $j$, the output of filter $j$ is given by,

$$x_j(k+1) = \mu x_i(k) + (1-\mu)x_j(k).$$

In this paper we only consider the case where $\mu$ is fixed, although better results can be obtained if it is adaptive.

Networks that have hidden dynamics have states which are not directly accessible to observation. In fact, it may be impossible to determine the states of a system from observations of it's inputs and outputs alone. We divide networks with hidden dynamics into three classes: single layer networks, multilayer networks, and networks with local feedback.

Single layer networks are perhaps the most popular of the recurrent neural network models. In a single layer network, every node depends on the previous output of all of the other nodes. The function performed by each node distinguishes the types of recurrent networks in this class. In each of the networks, nodes can be characterized as a nonlinear function of a weighted sum of inputs, previous node outputs, or products of these values. A bias term may also be included. In this paper we consider first–order networks, high–order networks [5], bilinear networks, and Quadratic networks[12]. The terms that are weighted in each of these networks are summarized in Table 1.

Multilayer networks consist of a feedforward network coupled with a finite set of delays as shown in Figure 1b. One network in this class is an architecture proposed by Robinson and Fallside [11], in which the feedforward network is an MLP. Another popular networks that fits into this class is Elman's Simple Recurrent Network (SRN) [3]. An Elman network can be thought of as a single layer network with an extra layer of nodes that compute the output function, as shown in Figure 1c.

In locally recurrent networks the feedback is provided locally within each individual

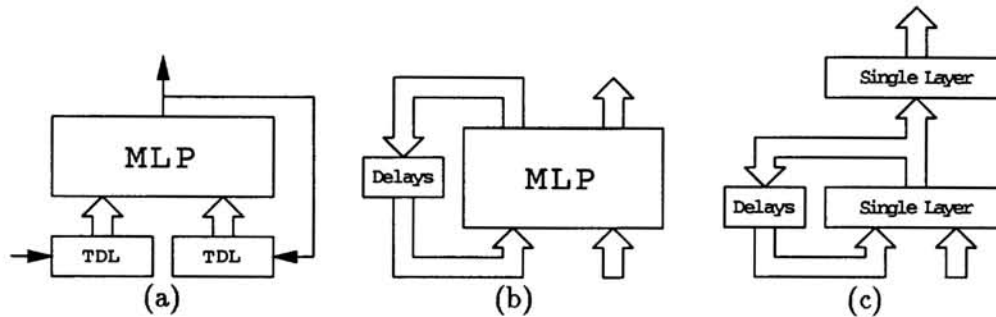

Figure 1: Network architectures: (a) Narendra and Parthasarathy's Recurrent Neural Network, (b) Multilayer network and (c) an Elman network.

node, but the nodes are connected together in a feed forward architecture. Specifically, we consider nodes that have local output feedback in which each node weights a window of its own past outputs and windows of node outputs from previous layers. Networks with local recurrence have been proposed in [1, 4, 10].

## 3   Experimental Results

### 3.1   Experimental methodology

In order to make the comparison as fair as possible we have adopted the following methodology.

- **Resources.** We shall perform two fundamental comparisons. One in which the number of weights is roughly the same for all networks, another in which the number of states is equivalent. In either case, we shall make these numbers large enough that most of the networks can achieve interesting performance levels.

  *Number of weights.* For static networks it is well known that the generalization performance is related to the number of weights in the network. Although this theory has never been extended to recurrent neural networks, it seems reasonable that a similar result might apply. Therefore, in some experiments we shall try to keep the number of weights approximately equal across all networks.

  *Number of states.* It can be argued that for dynamic problems the size of the state space is a more relevant measure for comparison than the number of weights. Therefore, in some experiments we shall keep the number of states equal across all networks.

- **Vanilla learning.** Several heuristics have been proposed to help speed learning and improve generalization of gradient descent learning algorithms. However, such heuristics may favor certain architectures. In order to avoid these issues, we have chosen simple gradient descent learning algorithms.

- **Number of simulations.** Due to random initial conditions, the recurrent neural network solutions can vary widely. Thus, to try to achieve a statistically significant estimation of the generalization of these networks, a large number of experiments were run.

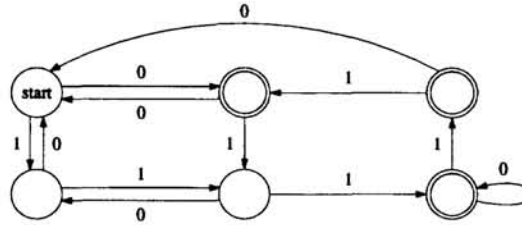

Figure 2: A randomly generated six state finite state machine.

## 3.2 Finite state machines

We chose two finite state machine (FSM) problems for a comparison of the ability of the various recurrent networks to perform grammatical inference. The first problem is to learn the minimal, randomly generated six state machine shown in Figure 2. The second problem is to infer a sixty–four state finite memory machine [6] described by the logic function

$$y(k) = \bar{u}(k-3)\bar{u}(k) + \bar{u}(k-3)y(k-3) + u(k)u(k-3)\bar{y}(k-3)$$

where $u(k)$ and $y(k)$ represent the input and output respectively at time $k$ and $\bar{x}$ represents the complement of $x$.

Two experiments were run. In the first experiment all of the networks were designed such that the number of weights was less than, but as close to 60 as possible. In the second experiment, each network was restricted to six state variables, and if possible, the networks were designed to have approximately 75 weights. Several alternative architectures were tried when it was possible to configure the architecture differently and yield the same number of weights, but those used gave the best results.

A complete set of 254 strings consisting of all strings of length one through seven is sufficient to uniquely identify both of these FSMs. For each simulation, we randomly partitioned the data into a training and testing set consisting of 127 strings each. The strings were ordered lexographically in the training set.

For each architecture 100 runs were performed on each problem. The on–line Back Propagation Through Time (BPTT) algorithm was used to train the networks. Vanilla learning was used with a learning rate of 0.5. Training was stopped at 1000 epochs. The weights of all networks were initialized to random values uniformly distributed in the range $[-0.1, 0.1]$. All states were initialize to zeros at the beginning of each string except for the High Order net in which one state was arbitrarily initialized to a value of 1.

Table 2 summarizes the statistics for each experiment. From these results we draw the following conclusions.

- The bilinear and high–order networks do best on the small randomly generated machine, but poorly on the finite memory machine. Thus, it would appear that there is benefit to having second order terms in the network, at least for small finite state machine problems.
- Narendra and Parthasarathy's model and the network with local recurrence do far better than the other networks on the problem of inferring the finite memory

Table 2: Percentage classification error on the FSM experiment for (a) networks with approximately the same number of weights, (b) networks with the same number of state variables. %P = The percentage of trials in which the training set was learned perfectly, #W = the number of weights, and #S = the number of states.

| FSM | Architecture[†] | training error mean | (std) | testing error mean | (std) | % P | # W | #S |
|-----|-----------------|---------------------|-------|--------------------|-------|-----|-----|-----|
| | N & P | 2.8 | (4.4) | 16.9 | (8.6) | 22 | 56 | 8 |
| | TDNN | 12.5 | (2.1) | 33.8 | (4.1) | 0 | 56 | 8 |
| | Gamma | 19.6 | (2.4) | 24.8 | (3.2) | 0 | 56 | 8 |
| | First Order | 12.9 | (6.9) | 26.5 | (9.0) | 0 | 48 | 6 |
| RND | High Order | 0.8 | (1.5) | 6.2 | (6.1) | 60 | 50 | 5 |
| | Bilinear | 1.3 | (2.7) | 5.7 | (6.1) | 46 | 55 | 5 |
| | Quadratic | 12.9 | (13.4) | 17.7 | (14.1) | 12 | 45 | 3 |
| | Multilayer | 19.4 | (13.6) | 23.4 | (13.5) | 6 | 54 | 4 |
| | Elman | 3.5 | (5.5) | 12.7 | (9.1) | 27 | 55 | 6 |
| | Local | 2.8 | (1.5) | 26.7 | (7.6) | 4 | 60 | 20 |
| | N & P | 0.0 | (0.2) | 0.1 | (1.1) | 99 | 56 | 8 |
| | TDNN | 6.9 | (2.1) | 15.8 | (3.2) | 0 | 56 | 8 |
| | Gamma | 7.7 | (2.2) | 15.7 | (3.3) | 0 | 56 | 8 |
| | First Order | 4.8 | (3.0) | 16.0 | (6.5) | 1 | 48 | 6 |
| FMM | High Order | 5.3 | (4.0) | 26.0 | (5.1) | 1 | 50 | 5 |
| | Bilinear | 9.5 | (10.4) | 25.8 | (7.0) | 0 | 55 | 5 |
| | Quadratic | 32.5 | (10.8) | 40.5 | (7.3) | 0 | 45 | 3 |
| | Multilayer | 36.7 | (11.9) | 43.5 | (8.5) | 0 | 54 | 4 |
| | Elman | 12.0 | (12.5) | 24.9 | (7.9) | 5 | 55 | 6 |
| | Local | 0.1 | (0.3) | 1.0 | (3.0) | 97 | 60 | 20 |

(a)

| FSM | Architecture[††] | training error mean | (std) | testing error mean | (std) | % P | # W | #S |
|-----|------------------|---------------------|-------|--------------------|-------|-----|-----|-----|
| | N & P | 4.6 | (8.4) | 14.1 | (11.3) | 38 | 73 | 6 |
| | TDNN | 11.7 | (2.0) | 34.3 | (3.9) | 0 | 73 | 6 |
| | Gamma | 19.0 | (2.4) | 25.2 | (3.1) | 0 | 74 | 6 |
| | First Order | 12.9 | (6.9) | 26.5 | (9.0) | 0 | 48 | 6 |
| RND | High Order | 0.3 | (0.5) | 4.6 | (5.1) | 79 | 74 | 6 |
| | Bilinear | 0.6 | (0.9) | 4.4 | (4.6) | 55 | 78 | 6 |
| | Quadratic | 0.2 | (0.5) | 3.2 | (2.6) | 83 | 216 | 6 |
| | Multilayer | 15.4 | (14.1) | 19.9 | (14.4) | 16 | 76 | 6 |
| | Elman | 3.5 | (5.5) | 12.7 | (9.1) | 27 | 55 | 6 |
| | Local | 13.9 | (4.5) | 20.2 | (5.7) | 0 | 26 | 6 |
| | N & P | 0.1 | (0.8) | 0.3 | (1.4) | 97 | 73 | 6 |
| | TDNN | 6.8 | (1.7) | 16.2 | (2.9) | 0 | 73 | 6 |
| | Gamma | 9.0 | (2.9) | 14.9 | (2.8) | 0 | 73 | 6 |
| | First Order | 4.8 | (3.0) | 16.0 | (6.5) | 1 | 48 | 6 |
| FMM | High Order | 1.2 | (1.7) | 25.1 | (5.1) | 31 | 74 | 6 |
| | Bilinear | 2.6 | (4.2) | 20.3 | (7.2) | 21 | 78 | 6 |
| | Quadratic | 12.6 | (17.3) | 26.1 | (12.8) | 13 | 216 | 6 |
| | Multilayer | 38.1 | (12.6) | 42.8 | (9.2) | 0 | 76 | 6 |
| | Elman | 12.8 | (14.8) | 27.6 | (10.7) | 8 | 55 | 6 |
| | Local | 15.3 | (3.8) | 22.2 | (4.9) | 0 | 26 | 6 |

(b)

[†]The TDNN and Gamma network both had 8 input taps and 4 hidden layer nodes. For the Gamma network, $\mu = 0.3$ (RND) and $\mu = 0.7$ (FMM). Narendra and Parthasarathy's network had 4 input and output taps and 5 hidden layer nodes. The High-order network used a "one-hot" encoding of the input values [5]. The multilayer network had 4 hidden and output layer nodes. The locally recurrent net had 4 hidden layer nodes with 5 input and 3 output taps, and one output node with 3 input and output taps.

[††]The TDNN, Gamma network, and Narendra and Parthasarathy's network all had 8 hidden layer nodes. For the Gamma network, $\mu = 0.3$ (RND) and $\mu = 0.7$ (FMM). The High-order network again used a "one-hot" encoding of the input values. The multilayer network had 5 hidden and 6 output layer nodes. The locally recurrent net had 3 hidden layer nodes and one output layer node, all with only one input and output tap.

machine when the number of states is not constrained. It is not surprising that the former network did so well since the sequential machine implementation of a finite memory machine is similar to this architecture [6]. However, the result for the locally recurrent network was unexpected.

- All of the recurrent networks do better than the TDNN on the small random machine. However, on the finite memory machine the TDNN does surprisingly well, perhaps because its structure is similiar to Narendra and Parthasarathy's network which was well suited for this problem.

- Gradient–based learning algorithms are not adequate for many of these architectures. In many cases a network is capable of representing a solution to a problem that the algorithm was not able to find. This seems particularly true for the Multilayer network.

- Not surprisingly, an increase in the number of weights typically leads to overtraining. Although, the quadratic network, which has 216 weights, can consistently find solutions for the random machine that generalize well even though there are only 127 training samples.

- Although the performance on the training set is not always a good indicator of generalization performance on the testing set, we find that if a network is able to frequently find perfect solutions for the training data, then it also does well on the testing data.

### 3.3  Nonlinear system identification

In this problem, we train the network to learn the dynamics of the following set of equations proposed in [8]

$$x_1(k+1) = \frac{x_1(k) + 2x_2(k)}{1 + x_2^2(k)} + u(k)$$

$$x_2(k+1) = \frac{x_1(k)x_2(k)}{1 + x_2^2(k)} + u(k)$$

$$y(k) = x_1(k) + x_2(k)$$

based on observations of $u(k)$ and $y(k)$ alone.

The same networks that were used for the finite state machine problems were used here, except that the output node was changed to be linear instead of sigmoidal to allow the network to have an appropriate dynamic range. We found that this caused some stability problems in the quadratic and locally recurrent networks. For the fixed number of weights comparison, we added an extra node to the quadratic network, and dropped any second order terms involving the fed back output. This gave a network with 64 weights and 4 states. For the fixed state comparison, dropping the second order terms gave a network with 174 weights. The locally recurrent network presented stability problems only for the fixed number of weights comparison. Here, we used a network that had 6 hidden layer nodes and one output node with 2 taps on the inputs and outputs each, giving a network with 57 weights and 16 states. In the Gamma network a value of $\mu = 0.8$ gave the best results.

The networks were trained with 100 uniform random noise sequences of length 50. Each experiment used a different randomly generated training set. The noise was

Table 3: Normalized mean squared error on a sinusoidal test signal for the nonlinear system identification experiment.

| Architecture | Fixed # weights | Fixed # states |
|---|---|---|
| N & P | 0.101 | 0.067 |
| TDNN | 0.160 | 0.165 |
| Gamma | 0.157 | 0.151 |
| First Order | 0.105 | 0.105 |
| High Order | 1.034 | 1.050 |
| Bilinear | 0.118 | 0.111 |
| Quadratic | 0.108 | 0.096 |
| Multilayer | 0.096 | 0.084 |
| Elman | 0.115 | 0.115 |
| Local | 0.117 | 0.123 |

uniformly distributed in the range $[-2.0, 2.0]$, and each sequence started with an initial value of $x_1(0) = x_2(0) = 0$. The networks were tested on the response to a sine wave of frequency 0.04 radians/second. This is an interesting test signal because it is fundamentally different than the training data.

Fifty runs were performed for each network. BPTT was used for 500 epochs with a learning rate of 0.002. The weights of all networks were initialized to random values uniformly distributed in the range $[-0.1, 0.1]$.

Table 3 shows the normalized mean squared error averaged over the 50 runs on the testing set. From these results we draw the following conclusions.

- The high order network could not seem to match the dynamic range of its output to the target, as a result it performed much worse than the other networks. It is clear that there is benefit to adding first order terms since the bilinear network performed so much better.

- Aside from the high order network, all of the other recurrent networks performed better than the TDNN, although in most cases not significantly better.

- The multilayer network performed exceptionally well on this problem, unlike the finite state machine experiments. We speculate that the existence of target output at every point along the sequence (unlike the finite state machine problems) is important for the multilayer network to be successful.

- Narendra and Parthasarathy's architecture did exceptionally well, even though it is not clear that its structure is well matched to the problem.

## 4   Conclusions

We have reviewed many discrete–time recurrent neural network architectures and compared them on two different problem domains, although we make no claim that any of these results will necessarily extend to other problems.

Narendra and Parthasarathy's model performed exceptionally well on the problems we explored. In general, single layer networks did fairly well, however it is important to include terms besides simple state/input products for nonlinear system identification. All of the recurrent networks usually did better than the TDNN except

on the finite memory machine problem. In these experiments, the use of averaging filters as a substitute for taps in the TDNN did not seem to offer any distinct advantages in performance, although better results might be obtained if the value of $\mu$ is adapted.

We found that the relative comparison of the networks did not significantly change whether or not the number of weights or states were held constant. In fact, holding one of these values constant meant that in some networks the other value varied wildly, yet there appeared to be little correlation with generalization.

Finally, it is interesting to note that though some are much better than others, many of these networks are capable of providing adequate solutions to two seemingly disparate problems.

## Acknowledgements

We would like to thank Leon Personnaz and Isabelle Rivals for suggesting we perform the experiments with a fixed number of states.

## Footnotes

*Also with UMIACS, University of Maryland, College Park, MD 20742

## References

[1] A.D. Back and A.C. Tsoi. FIR and IIR synapses, a new neural network architecture for time series modeling. *Neural Computation*, 3(3):375–385, 1991.

[2] B. de Vries and J.C. Principe. The gamma model: A new neural model for temporal processing. *Neural Networks*, 5:565–576, 1992.

[3] J.L. Elman. Finding structure in time. *Cognitive Science*, 14:179–211, 1990.

[4] P. Frasconi, M. Gori, and G. Soda. Local feedback multilayered networks. *Neural Computation*, 4:120–130, 1992.

[5] C.L. Giles, C.B. Miller, et al. Learning and extracting finite state automata with second–order recurrent neural networks. *Neural Computation*, 4:393–405, 1992.

[6] Z. Kohavi. *Switching and finite automata theory*. McGraw–Hill, NY, 1978.

[7] K.J. Lang, A.H. Waibel, and G.E. Hinton. A time–delay neural network architecture for isolated word recognition. *Neural Networks*, 3:23–44, 1990.

[8] K.S. Narendra. Adaptive control of dynamical systems using neural networks. In *Handbook of Intelligent Control*, pages 141–183. Van Nostrand Reinhold, NY, 1992.

[9] K.S. Narendra and K. Parthasarathy. Identification and control of dynamical systems using neural networks. *IEEE Trans. on Neural Networks*, 1:4–27, 1990.

[10] P. Poddar and K.P. Unnikrishnan. Non–linear prediction of speech signals using memory neuron networks. In *Proc. 1991 IEEE Work. Neural Networks for Sig. Proc.*, pages 1–10. IEEE Press, 1991.

[11] A.J. Robinson and F. Fallside. Static and dynamic error propagation networks with application to speech coding. In *NIPS*, pages 632–641, NY, 1988. AIP.

[12] R.L. Watrous and G.M. Kuhn. Induction of finite–state automata using second–order recurrent networks. In *NIPS4*, pages 309–316, 1992.